# Short-term memory in neuronal networks through dynamical compressed sensing

**Surya Ganguli**

Sloan-Swartz Center for Theoretical Neurobiology, UCSF, San Francisco, CA 94143
surya@phy.ucsf.edu

**Haim Sompolinsky**

Interdisciplinary Center for Neural Computation, Hebrew University, Jerusalem 91904, Israel
and Center for Brain Science, Harvard University, Cambridge, Massachusetts 02138, USA
haim@fiz.huji.ac.il

## Abstract

Recent proposals suggest that large, generic neuronal networks could store memory traces of past input sequences in their instantaneous state. Such a proposal raises important theoretical questions about the duration of these memory traces and their dependence on network size, connectivity and signal statistics. Prior work, in the case of gaussian input sequences and linear neuronal networks, shows that the duration of memory traces in a network cannot exceed the number of neurons (in units of the neuronal time constant), and that no network can out-perform an equivalent feedforward network. However a more ethologically relevant scenario is that of sparse input sequences. In this scenario, we show how linear neural networks can essentially perform compressed sensing (CS) of past inputs, thereby attaining a memory capacity that *exceeds* the number of neurons. This enhanced capacity is achieved by a class of "orthogonal" recurrent networks and not by feedforward networks or generic recurrent networks. We exploit techniques from the statistical physics of disordered systems to analytically compute the decay of memory traces in such networks as a function of network size, signal sparsity and integration time. Alternately, viewed purely from the perspective of CS, this work introduces a new ensemble of measurement matrices derived from dynamical systems, and provides a theoretical analysis of their asymptotic performance.

## 1 Introduction

How neuronal networks can store a memory trace for recent sequences of stimuli is a central question in theoretical neuroscience. The influential idea of attractor dynamics [1], suggests how single stimuli can be stored as stable patterns of activity, or fixed point attractors, in the dynamics of recurrent networks. But, such simple fixed points are incapable of storing sequences. More recent proposals [2, 3, 4] suggest that recurrent networks could store temporal sequences of inputs in their ongoing, transient activity, even if they do not have nontrivial fixed points. In principle, past inputs could be read out from the instantaneous activity of the network. However, the theoretical principles underlying the ability of recurrent networks to store temporal sequences in their transient dynamics are poorly understood. For example, how long can memory traces last in such networks, and how does memory capacity depend on parameters like network size, connectivity, or input statistics?

Several recent theoretical studies have made progress on these issues in the case of linear neuronal networks and gaussian input statistics. Even in this simple setting, the relationship between the memory properties of a neural network and its connectivity is nonlinear, and so understanding this

relationship poses an interesting challenge. Jaeger [4] proved a rigorous sum-rule (reviewed in more detail below) which showed that even in the absence of noise, no recurrent network can remember inputs for an amount of time that exceeds the number of neurons (in units of the neuronal time constant) in the network. White et al. [5] showed that in the presence of noise, a special class of "orthogonal" networks, but not generic recurrent networks, could have memory that scales with network size. And finally, Ganguli et. al. [6] used the theory of Fisher information to show that the memory of a recurrent network cannot exceed that of an equivalent feedforward network, at least for times up to the network size, in units of the neuronal time constant.

A key reason theoretical progress was possible in these works was that even though the optimal estimate of past inputs was a nonlinear function of the network connectivity, it was still a linear function of the current network state, due to the gaussianity of the signal (and possible noise) and the linearity of the dynamics. It is not clear for example, how these results would generalize to nongaussian signals, whose reconstruction from the current network state would require nonlinear operations. Here we report theoretical progress on understanding the memory capacity of linear recurrent networks for an important class of nongaussian signals, namely sparse signals. Indeed a wide variety of temporal signals of interest are sparse in some basis, for example human speech in a wavelet basis. We use ideas from compressed sensing (CS) to define memory curves which capture the decay of memory traces in neural networks for sparse signals, and provide methods to compute these curves analytically. We find strikingly different properties of memory curves in the sparse setting compared to the gaussian setting. Although motivated by the problem of memory, we also contribute new results to the field of CS itself, by introducing and analyzing new classes of CS measurement matrices derived from dynamical systems. Our main results are summarized in the discussion section. In the next section, we begin by reviewing more quantitatively the problem of short-term memory in neuronal networks, compressed sensing, and the relation between the two.

## 2  Short-term memory as dynamical compressed sensing.

Consider a discrete time network dynamics given by

$$\mathbf{x}(n) = \mathbf{W}\mathbf{x}(n-1) + \mathbf{v}s^0(n). \tag{1}$$

Here a scalar, time dependent signal $s^0(n)$ drives a recurrent network of $N$ neurons. $\mathbf{x}(n) \in \mathcal{R}^N$ is the network state at time $n$, $\mathbf{W}$ is an $N \times N$ recurrent connectivity matrix, and $\mathbf{v}$ is a vector of feedforward connections from the signal into the network. We choose $\mathbf{v}$ to have norm 1, and we demand that the dynamics be stable so that if $\rho$ is the squared magnitude of the largest eigenvalue of $\mathbf{W}$, then $\rho < 1$. If we think of the signal history $\{s^0(n-k)|k \geq 0\}$ as an infinite dimensional temporal vector $\mathbf{s}^0$ whose $k$'th component $\mathbf{s}_k^0$ is $s(n-k)$, then the current network state $\mathbf{x}$ is linearly related to $\mathbf{s}$ through the effective $N$ by $\infty$ measurement matrix $\mathbf{A}$, i.e. $\mathbf{x} = \mathbf{A}\mathbf{s}^0$, where the matrix elements

$$\mathbf{A}_{\mu k} = (\mathbf{W}^k \mathbf{v})_\mu, \quad \mu = 1, \dots, N, \ \ k = 0, \dots, \infty \tag{2}$$

reflect the effect of an input $k$ timesteps in the past on the activity of neuron $\mu$. The extent to which the dynamical system in (1) can remember the past can then be quantified by how well one can recover $\mathbf{s}^0$ from $\mathbf{x}$ [4, 5, 6]. In the case where the signal has zero mean gaussian statistics with covariance $\langle \mathbf{s}_k^0 \mathbf{s}_l^0 \rangle = \delta_{k,l}$, the optimal, minimum mean squared error estimate $\hat{\mathbf{s}}$ of the signal history is given by $\hat{\mathbf{s}} = \mathbf{A}^T (\mathbf{A}\mathbf{A}^T)^{-1} \mathbf{x}$. The correlation between the estimate $\hat{\mathbf{s}}_k$ and the true signal $\mathbf{s}_k^0$, averaged over the gaussian statistics of $\mathbf{s}^0$, then defines a memory curve $M(k) = \langle \hat{\mathbf{s}}_k \mathbf{s}_k^0 \rangle_{\mathbf{s}^0}$, whose decay as $k$ increases quantifies the decay of memory for past inputs in (1). Jaeger proved an important sum-rule for $M(k)$: $\sum_{k=0}^\infty M(k) = N$ for any recurrent connectivity $\mathbf{W}$ and feedforward connectivity $\mathbf{v}$. Given that $M(k)$ cannot exceed 1 for any $k$, an important consequence of this sum-rule is that it is not possible to recover an input signal $k$ timesteps into the past when $k$ is much larger than $N$ in the sense that $\hat{\mathbf{s}}_k$ will be at most weakly correlated with $\mathbf{s}_k^0$.

Generically, one may not hope to remember sequences lasting longer than $N$ timesteps with only $N$ neurons, but in the case of temporally sparse inputs, the field of compressed sensing (CS) suggests this may be possible. CS [7, 8] shows how to recover a sparse $T$ dimensional signal $\mathbf{s}^0$, in which only a fraction $f$ of the elements are nonzero, from a set of $N$ linear measurements $\mathbf{x} = \mathbf{A}\mathbf{s}^0$ where $\mathbf{A}$ is an $N$ by $T$ measurement matrix with $N < T$. One approach to recovering an estimate $\hat{\mathbf{s}}$ of $\mathbf{s}^0$

from $\mathbf{x}$ involves $L_1$ minimization,

$$\hat{\mathbf{s}} = \arg \min_{\mathbf{s}} \sum_{i=1}^{T} |s_i| \quad \text{subject to } \mathbf{x} = \mathbf{A}\mathbf{s}, \tag{3}$$

which finds the sparsest signal, as measured by smallest $L_1$ norm, consistent with the measurement constraints. Much of the seminal work in CS [9, 10, 11] has focused on sufficient conditions on $\mathbf{A}$ such that (3) is guaranteed to perfectly recover the true signal, so that $\hat{\mathbf{s}} = \mathbf{s}^0$. However, many large random measurement matrices $\mathbf{A}$ which violate sufficient conditions proven in the literature still nevertheless typically yield perfect signal recovery. Alternate work [12, 13, 14, 15] which analyzes the asymptotic performance of large random measurement matrices in which each matrix element is drawn i.i.d. from a gaussian distribution, has revealed a phase transition in performance as a function the signal sparsity $f$ and the degree of subsampling $\alpha = N/T$. In the $\alpha$-$f$ plane, there is a critical phase boundary $\alpha_c(f)$ such that if $\alpha > \alpha_c(f)$ then CS will typically yield perfect signal reconstruction, whereas if $\alpha < \alpha_c(f)$, CS will yield errors.

Motivated by the above work in CS, we propose here that a neural network, or more generally any dynamical system as in (1), could in principle perform compressed sensing of its past inputs, and that a long but sparse signal history $\mathbf{s}^0$ could potentially be recovered from the instantaneous network state $\mathbf{x}$. We quantify the memory capabilities of a neural network for sparse signals, by assessing our ability to reconstruct the past signal using $L_1$ minimization. Given a network state $\mathbf{x}$ arising from a signal history $\mathbf{s}^0$ through (1), we can obtain an estimate $\hat{\mathbf{s}}$ of the past using (3), where the measurement matrix $\mathbf{A}$ is given by (2). We then define a memory curve

$$E(k) = \langle (\hat{\mathbf{s}}_k - \mathbf{s}_k^0)^2 \rangle_{\mathbf{s}^0}, \tag{4}$$

namely the average reconstruction error of a signal $k$ timesteps in the past averaged over the statistics of $\mathbf{s}^0$. The rise of this error as $k$ increases captures the decay of memory traces in (1). The central goal of this paper is to obtain a deeper understanding of the memory properties of neural networks for sparse signals by studying the memory curve $E(k)$ and especially its dependence on $\mathbf{W}$. In particular, we are interested in classes of network connectivities $\mathbf{W}$ and input statistics for which $E(k)$ can remain small even for $k \gg N$. Such networks can essentially perform compressed sensing of their past inputs.

From the perspective of CS, measurement matrices $\mathbf{A}$ of the form in (2), henceforth referred to as dynamical CS matrices, possess several new features not considered in the existing CS literature, features which could pose severe challenges for a recurrent network $\mathbf{W}$ to achieve good CS performance. First, $\mathbf{A}$ is an $N$ by $\infty$ matrix, and so from the perspective of the phase diagram for CS reviewed above, it is likely that $\mathbf{A}$ is in the error phase; thus perfect reconstruction of the true signal, even for recent inputs will not be possible. Second, because we demand stable dynamics in (1), the columns of $\mathbf{A}$ decay as $k$ increases: $||\mathbf{W}^k \mathbf{v}||^2 < \rho^k$ where again $\rho < 1$ is the squared magnitude of the largest eigenvalue of $\mathbf{W}$. Such decay can compound errors. Third, the different columns of $\mathbf{A}$ can be correlated; if one thinks of $\mathbf{W}^k \mathbf{v}$ as the state of the network $k$ timesteps after a single unit input pulse, it is clear that temporal correlations in the evolving network response to this pulse are equivalent to correlations in the columns of $\mathbf{A}$ in (2). Such correlations could potentially adversely affect the performance of CS based on $\mathbf{A}$, as well as complicate the theoretical analysis of CS performance. Nevertheless, despite all these seeming difficulties, in the following we show that a special class of network connectivities can indeed achieve good CS performance in which errors are controlled and memory traces can last longer than the number of neurons.

## 3 Memory in an Annealed Approximation to a Dynamical System

In this section, we work towards an analytic understanding of the memory curve $E(k)$ defined in (4). This curve depends on $\mathbf{W}$, $\mathbf{v}$ and the statistics of $\mathbf{s}^0$. We would like to understand its properties for ensembles of large random networks $\mathbf{W}$, just as the asymptotic performance of CS was analyzed for large random measurement matrices $\mathbf{A}$ [12, 13, 14, 15]. However, in the dynamical setting, even if $\mathbf{W}$ is drawn from a simple random matrix ensemble, $\mathbf{A}$ in (2) will have correlations across its columns, making an analytical treatment of the memory curve difficult. Here we consider an ensemble of measurement matrices $\mathbf{A}$ which approximate dynamical CS matrices and can be

treated analytically. We consider matrices in which each element $\mathbf{A}_{\mu k}$ is drawn i.i.d from a zero mean gaussian distribution with variance $\rho^k$. Since we are interested in memory that lasts $O(N)$ timesteps, we choose $\rho = e^{-1/\tau N}$, with $\tau$ $O(1)$. This so called annealed approximation (AA) to a dynamical CS matrix captures two of the salient properties of dynamical CS matrices, their infinite temporal extent and the decay of successive columns, but neglects the analytically intractable correlations across columns. Such annealed CS matrices can be thought of as arising from "imaginary" dynamical systems in which network activity patterns over time in response to a pulse decay, but are somehow temporally uncorrelated. $\tau$ can be thought of as the effective integration time of this dynamical system, in units of the number of neurons. Finally, to fully specify $E(k)$, we must choose the statistics of $\mathbf{s}^0$. We assume $\mathbf{s}^0$ has a probability $f$ of being nonzero at any given time, and if nonzero, this nonzero value is drawn from a distribution $P(s)$ which for now we take to be arbitrary.

To theoretically compute the memory curve $E(k)$, we define an energy function

$$E(\mathbf{s}) = \frac{\lambda}{2}\mathbf{u}^T\mathbf{A}^T\mathbf{A}\mathbf{u} + \sum_{i=1}^{T}|\mathbf{s}_i|, \tag{5}$$

where $\mathbf{u} \equiv \mathbf{s} - \mathbf{s}^0$ is the residual, and we consider the Gibbs distribution $P_G(\mathbf{s}) = \frac{1}{Z}e^{-\beta E(\mathbf{s})}$. We will later take $\lambda \to \infty$ so that the quadratic part of the energy function enforces the constraint $\mathbf{A}\mathbf{s} = \mathbf{A}\mathbf{s}^0$, and then take the low temperature $\beta \to \infty$ limit so that $P_G(\mathbf{s})$ concentrates onto the global minimum of (3). In this limit, we can extract the memory curve $E(k)$ as the average of $(\mathbf{s}_k - \mathbf{s}_k^0)^2$ over $P_G$ and the statistics of $\mathbf{s}^0$. Although $P_G$ depends on $\mathbf{A}$, for large $N$, the properties of $P_G$, including the memory curve $E(k)$, do not depend on the detailed realization of $\mathbf{A}$, but only on its statistics. Indeed we can compute all properties of $P_G$ for any typical realization of $\mathbf{A}$ by averaging over both $\mathbf{A}$ and $\mathbf{s}^0$. This is done using the replica method [16] in our supplementary material. The replica method has been used recently in several works to analyze CS for the traditional case of uniform random gaussian measurement matrices [14, 17, 15]. We find that the statistics of each component $s_k$ in $P_G(\mathbf{s})$, conditioned on the true value $s_k^0$ is well described by a mean field effective Hamiltonian

$$H_k^{MF}(s) = \rho^k \frac{\beta\lambda}{2(1 + \beta\lambda\Delta Q)}\left(s - s_k^0 - z\sqrt{Q_0/\rho^k}\right)^2 + \beta|s|, \tag{6}$$

where $z$ is a random variable with a standard normal distribution. Thus the mean field approximation to the marginal distribution of a reconstruction component $s_k$ is

$$P_k^{MF}(s_k = s) = \int \mathcal{D}z \frac{1}{Z_k^{MF}}\exp(-H_k^{MF}(s)), \tag{7}$$

where $\mathcal{D}z = dz\,e^{-\frac{1}{2}z^2}$ is a Gaussian measure. The order parameters $Q_0$ and $\Delta Q \equiv Q_1 - Q_0$ obey

$$Q_0 = \frac{1}{N}\sum_{k=0}^{\infty}\rho^k\langle\langle\,\langle u\rangle_{H_k^{MF}}^2\,\rangle\rangle_z \tag{8}$$

$$\Delta Q = \frac{1}{N}\sum_{k=0}^{\infty}\rho^k\langle\langle\,\langle \delta u^2\rangle_{H_k^{MF}}\,\rangle\rangle_z. \tag{9}$$

Here $\langle u\rangle_{H_k^{MF}}$ and $\langle \delta u^2\rangle_{H_k^{MF}}$ are the mean and variance of the residual $u_k = s_k - s_k^0$ with respect to a Gibbs distribution with Hamiltonian given by (6), and the double angular average $\langle\langle\,\cdot\,\rangle\rangle_z$ refers to integrating over the Gaussian distribution of $z$. $Q_1$ and $Q_0$ have simple interpretations in terms of the original Gibbs distribution $P_G$ defined above: $Q_1 = \frac{1}{N}\sum_{k=1}^{\infty}\rho^k\langle u_k^2\rangle_{P_G}$ and $Q_0 = \frac{1}{N}\sum_{k=1}^{\infty}\rho^k\langle u_k\rangle_{P_G}^2$, for typical realizations of $\mathbf{A}$. Thus the order parameter equations (8)-(9) can be understood as self-consistency conditions for the definition of $Q_0$ and $\Delta Q$ in the mean field approximation to $P_G$. In this approximation, the complicated constraints coupling $s_k$ for various $k$ are replaced with a random gaussian force $z$ in (6) which tends to prevent the marginal $s_k$ from assuming the true value $s_k^0$. This force is what remains of the measurement constraints after averaging over $\mathbf{A}$, and its statistics are in turn a function of $Q_0$ and $Q_1$, as determined by the replica method.

Now to compute the memory curve $E(k)$, we must take the limits $\lambda, \beta, N \to \infty$ and complete the average over $\mathbf{s}_k^0$. The $\lambda \to \infty$ limit can be taken immediately in (6) and $\lambda$ disappears from the problem. Now as $\beta \to \infty$, self consistent solutions to (8) and (9) can be found when $Q_0 \equiv q_0$ and

$\Delta Q \equiv \Delta q/\beta$, where $q_0$ and $\Delta q$ are $O(1)$. This limit is similar to that taken in a replica analysis of CS for random gaussian matrices in the error regime [15]. Taking this limit, (6) becomes

$$H_k^{MF}(s) = \beta \left[ \frac{1}{2\rho^{-k}\Delta q} \left( s - s_k^0 - z\sqrt{\rho^{-k}q_0} \right)^2 + |s| \right]. \tag{10}$$

Since the entire Hamiltonian is proportional to $\beta$, in the large $\beta$ limit, the statistics of $s_k$ are dominated by the global minimum of (10). In particular, we have

$$\langle s \rangle_{H_k^{MF}} = \eta \left( s_k^0 + z\sqrt{\rho^{-k}q_0}, \rho^{-k}\Delta q \right), \tag{11}$$

where

$$\eta(x, \sigma) = \arg\min_s \left( \frac{1}{2} \frac{(s-x)^2}{\sigma} + |s| \right) = \operatorname{sgn}(x)(|x| - \sigma)_+, \tag{12}$$

is a soft thresholding function which also arises in message passing approaches [18] to solving the CS problem in (3), and $(y)_+ = y$ if $y > 0$ and is otherwise 0. The optimization in (12) can be understood intuitively as follows: suppose one measures a scalar value $x$ which is a true signal $s^0$ corrupted by additive gaussian noise with variance $\sigma$. Under a Laplace prior $e^{-|s^0|}$ on the true signal, $\eta(x, \sigma)$ is simply the MAP estimate of $s^0$ given the data $x$, which basically chooses the estimate $s = 0$ unless the data exceeds the noise level $\sigma$. Thus we see that in (10), $\rho^{-k}\Delta q$ plays the role of an effective noise level which increases with time $k$. Also, the variance of $s$ at large $\beta$ is

$$\langle (\delta s)^2 \rangle_{H_k^{MF}} = \frac{1}{\beta} \chi \left( s_k^0 + z\sqrt{\rho^{-k}q_0}, \rho^{-k}\Delta q \right), \tag{13}$$

where

$$\chi(x, \sigma) = \sigma \, \Theta(|x| - \sigma), \tag{14}$$

and $\Theta(x)$ is a step function at 0. Inserting (11) and (13) and the ansatz $\Delta Q \equiv \Delta q/\beta$ into (8) and (9) then removes $\beta$ from the problem. But before making these substitutions, we first take $N \to \infty$ at fixed $\tau$ and $f$ of $O(1)$ by taking a continuum approximation for time, $t = k/N$, $\rho^k \to e^{-t/\tau}$, $\frac{1}{N} \sum_{k=0}^\infty \to \int_0^\infty dt$. Moreover, we average over the true signal history $s_k^0$, so that (8) and (9) become,

$$q_0 = \int_0^\infty dt \, e^{-t/\tau} \left\langle\!\left\langle \left( \eta(s^0 + z\sqrt{e^{t/\tau}q_0}, \, e^{t/\tau}\Delta q) - s^0 \right)^2 \right\rangle\!\right\rangle_{z,s^0} \tag{15}$$

$$\Delta q = \int_0^\infty dt \, e^{-t/\tau} \left\langle\!\left\langle \chi(s^0 + z\sqrt{e^{t/\tau}q_0}, \, e^{t/\tau}\Delta q) \right\rangle\!\right\rangle_{z,s^0}, \tag{16}$$

where the double angular average reflects an integral over the gaussian distribution of $z$ and the full distribution of $s^0$, i.e. $\left\langle\!\left\langle F(z, s^0) \right\rangle\!\right\rangle_{z,s^0} \equiv (1-f) \int \mathcal{D}z \, F(z, 0) + f \int \mathcal{D}z \, ds^0 \, P(s^0) F(z, s^0)$. Finally the memory curve $E(t)$ is simply the continuum limit of the averaged squared residual $\left\langle\!\left\langle \langle u \rangle_{H_k^{MF}}^2 \right\rangle\!\right\rangle_{z,s^0}$, and is given by

$$E(t) = \left\langle\!\left\langle \left( \eta(s^0 + z\sqrt{e^{t/\tau}q_0}, \, e^{t/\tau}\Delta q) - s^0 \right)^2 \right\rangle\!\right\rangle_{z,s^0}. \tag{17}$$

Equations (15),(16), and (17) now depend only on $\tau$, $f$ and $P(s^0)$, and their theoretical predictions can now be compared with numerical experiments. In this work we focus on a simple class of plus-minus (PM) signals in which $P(s^0) = 1/2 \, \delta(s^0 - 1) + 1/2 \, \delta(s^0 + 1)$. Fig. 1A shows an example of a PM signal $\mathbf{s}^0$ with $f = 0.01$, while Fig. 1B shows an example of a reconstruction of $\hat{\mathbf{s}}$ using $L_1$ minimization in (3) where the data $\mathbf{x}$ used in (3) was obtained from $\mathbf{s}^0$ using a random annealed measurement matrix with $\tau = 1$. Clearly there are errors in the reconstruction, but remarkably, despite the decay in the columns of $\mathbf{A}$, the reconstruction is well correlated with the true signal for a time up to 4 times the number of measurements. We can derive theoretical memory curves for any given $f$ and $\tau$ by numerically solving for $q_0$ and $\Delta q$ in (15),(16), and inserting the results into (17). Examples of the agreement between theory and simulations are shown in Fig. 1C-E.

As $t \to \infty$, $L_1$ minimization always yields a zero signal estimate, so the memory curve asymptotically approaches $f$ for large $t$. A convenient measure of memory capacity is the time $T_{1/2}$ at which the memory curve reaches half its asymptotic error value, i.e. $E(T_{1/2}) = f/2$. A principle feature

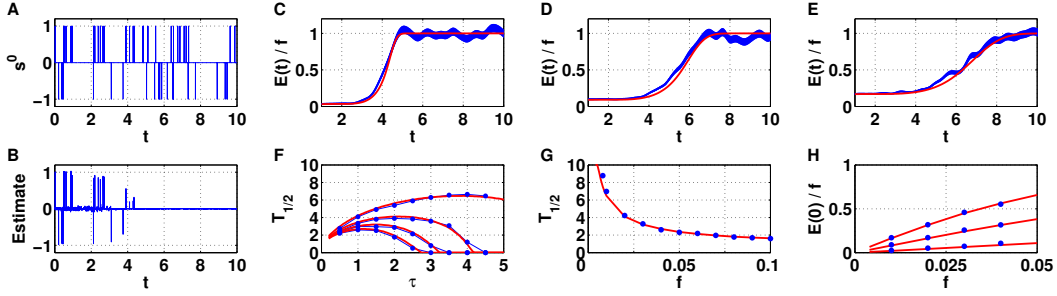

Figure 1: Memory in the annealed approximation. (A) A PM signal $\mathbf{s}^0$ with $f = 0.01$ that lasts $T = 10N$ timesteps where $N = 500$. (B) A reconstruction of $\mathbf{s}^0$ from the output of an annealed measurement matrix with $N = 500, \tau = 1$. (C,D,E) Example memory curves for $f = 0.01$, and $\tau = 1$ (C), 2 (D), 3 (E). (F) $T_{1/2}$ as a function of $\tau$. The 4 curves from top to bottom are for $f = 0.01, 0.02, 0.03, 0.04$. (G) $T_{1/2}$ optimized over $\tau$ for each $f$. (H) The initial error as a function of $f$. The 3 curves from bottom to top are for $\tau = 1, 2, 3$. For (C-H), red curves are theoretical predictions while blue curves and points are from numerical simulations of $L_1$ minimization with $N = 100$ averaged over 300 trials. The width of the blue curves reflects standard error.

of this family of memory curves is that for any given $f$ there is an optimal $\tau$ which maximizes $T_{1/2}$ (Fig. 1F) . The presence of this optimum arises due to a competition between decay and interference. If $\tau$ is too small, signal measurements decay too quickly, thereby preventing large memory capacity. However, if $\tau$ is too large, signals from the distant past do not decay away, thereby interfering with the measurements of more recent signals, and again degrading memory. As $f$ decreases, long time signal interference is reduced, thereby allowing larger values of $\tau$ to be chosen without degrading memory for more recent signals. For any given $f$, we can compute $T_{1/2}(f)$ optimized over $\tau$ (Fig. 1G). This memory capacity, again measured in units of the number of neurons, already exceeds 1 at modest values of $f = 0.1$, and diverges as $f \to 0$, as does the optimal value of $\tau$. By analyzing (15) and (16) in the limit $f \to 0$ and $\tau \to \infty$, we find that $\Delta q$ is $O(1)$ while $q_0 \to 0$. Furthermore, as $f \to 0$, the optimal $T_{1/2}$ is $O(\frac{1}{f \log 1/f})$.

The smallest error occurs at $t = 0$ and it is natural to ask how this error $E(0)$ behaves as a function of $f$ for small $f$ to see how well the most recent input can be reconstructed in the limit of sparse signals. We analyze (15) and (16) in the limit $f \to 0$ and $\tau$ of $O(1)$, and find that $E(0)$ is $O(f^2)$ as confirmed in Fig. 1F. Furthermore, $E(0)$ monotonically increases with $\tau$ for fixed $f$ as more signals from the past interfere with the most recent input.

## 4   Orthogonal Dynamical Systems

We have seen in the previous section that annealed CS matrices have remarkable memory properties, but our main interest was to exhibit a dynamical CS matrix as in (2) capable of good compressed sensing, and therefore short-term memory, performance. Here we show that a special class of network connectivity in which $\mathbf{W} = \sqrt{\rho}\mathbf{O}$ where $\mathbf{O}$ is any orthogonal matrix, and $\mathbf{v}$ is any random unit norm vector possesses memory properties remarkably close to that of the annealed matrix ensemble. Fig. 2A-F presents results identical to that of Fig. 1C-H except for the crucial change that all simulation results in Fig. 2 were obtained using dynamical CS matrices of the form $\mathbf{A}_{\mu k} = (\rho^{k/2}\mathbf{O}^k \mathbf{v})_\mu$, rather than annealed CS matrices. All red curves in Fig. 2A-F are identical to those in Fig. 1 and reflect the theory of annealed CS matrices derived in the previous section.

For small $\tau$, we see small discrepancies between memory curves for orthogonal neural networks and the annealed theory (Fig. 2A-B), but as $\tau$ increases, this discrepancy decreases (Fig. 2C). In particular, from the perspective of the optimal $T_{1/2}$ for which larger $\tau$ is relevant, we see a remarkable match between the optimal memory capacity of orthogonal neural networks and that predicted by the annealed theory (see Fig. 2E). And there is good match in the initial error even at small $\tau$ (Fig. 2F).

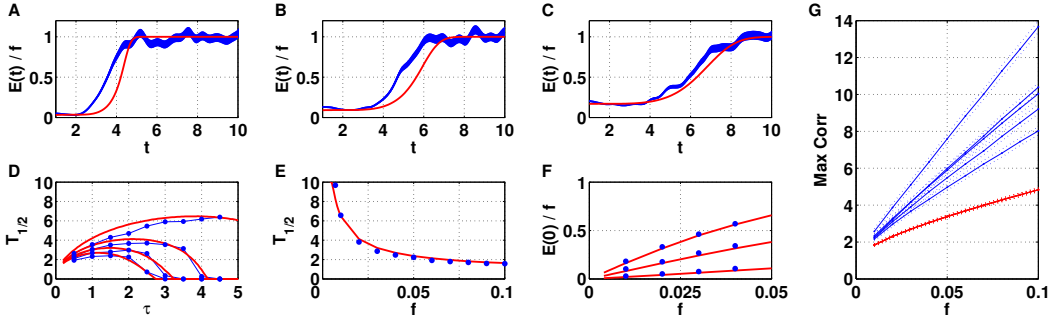

Figure 2: Memory in orthogonal neuronal networks. Panels (A-F) are identical to panels (C-H) in Fig. 1 except now the blue curves and points are obtained from simulations of $L_1$ minimization using measurement matrices derived from an orthogonal neuronal network. (G) The mean and standard deviation of $\sigma_f$ for 5 annealed (red) and 5 orthogonal matrices (blue) with N=200 and T=3000.

The key difference between the annealed and the dynamical CS matrices is that the former neglects correlations across columns that can arise in the latter. How strong are these correlations for the case of orthogonal matrices? Motivated by the restricted isometry property [11], we consider the following probe of the strength of correlations across columns of $\mathbf{A}$. Consider an $N$ by $fT$ matrix $\mathbf{B}$ obtained by randomly subsampling the columns of an $N$ by $T$ measurement matrix $\mathbf{A}$. Let $\sigma_f$ be the maximal eigenvalue of the matrix $\mathbf{B}^T\mathbf{B}$ of inner products of columns of $\mathbf{B}$. $\sigma_f$ is a measure of the strength of correlations across the $fT$ sampled columns of $\mathbf{A}$. We can estimate the mean and standard deviation of $\sigma_f$ due to the random choice of $fT$ columns of $\mathbf{A}$ and plot the results as function of $f$. To separate the issue of correlations from decay, we do this analysis for $\rho = 1$ and finite $T$ (similar results are obtained for large $T$ and $\rho < 1$). Results are shown in Fig 2 for 5 instances of annealed (red) and dynamical (blue) CS matrices. We see strikingly different behavior in the two ensembles. Correlations are much stronger in the dynamical ensemble, and fluctuate from instance to instance, while they are weaker in the annealed ensemble, and do not fluctuate (the 5 red curves are on top of each other). Given the very different statistical properties of the two ensembles, the level of agreement between the simulated memory properties of orthogonal neural networks, and the theory of annealed CS matrices is remarkable.

Why do orthogonal neural networks perform so well, and can more generic networks have similar performance? The key to understanding the memory, and CS, capabilities of orthogonal neural networks lies in the eigenvalue spectrum of an orthogonal matrix. The eigenvalues of $\mathbf{W} = \sqrt{\rho}\mathbf{O}$, when $\mathbf{O}$ is a large random orthogonal matrix, are uniformly distributed on a circle of radius $\sqrt{\rho}$ in the complex plane. Thus when $\rho = e^{-1/\tau N}$, the sequence of vectors $\mathbf{W}^k\mathbf{v}$ explore the full $N$ dimensional space of network activity patterns for $O(\tau N)$ time steps before decaying away. In contrast, a generic random Gaussian matrix $\mathbf{W}$ with elements drawn i.i.d from a zero mean gaussian with variance $\rho/N$ has eigenvalues uniformly distributed on a solid disk of radius $\sqrt{\rho}$ in the complex plane. Thus the sequence of vectors $\mathbf{W}^k\mathbf{v}$ no longer explore a high dimensional space of activity patterns; components of $\mathbf{v}$ in the direction of eigenmodes of $\mathbf{W}$ with small eigenvalues will rapidly decay away, and so the sequence will rapidly become confined to a low dimensional space. Good compressed sensing matrices often have columns that are random and uncorrelated. From the above considerations, it is clear that dynamical CS matrices derived from orthogonal neural networks can come close to this ideal, while those derived from generic gaussian networks cannot.

## 5    Discussion

In this work we have made progress on the theory of short-term memory for nongaussian, sparse, temporal sequences stored in the transient dynamics of neuronal networks. We used the framework of compressed sensing, specifically $L_1$ minimization, to reconstruct the history of the past input signal from the current network activity state. The reconstruction error as a function of time into the past then yields a well-defined memory curve that reflects the memory capabilities of the network. We studied the properties of this memory curve and its dependence on network connectivity, and found

results that were qualitatively different from prior theoretical studies devoted to short-term memory in the setting of gaussian input statistics. In particular we found that orthogonal neural networks, but importantly, not generic random gaussian networks, are capable of remembering inputs for a time that exceeds the number of neurons in the network, thereby circumventing a theorem proven in [4], which limits the memory capacity of any network to be less than the number of neurons in the gaussian signal setting. Also, recurrent connectivity plays an essential role in allowing a network to have a memory capacity that exceeds the number of neurons. Thus purely feedforward networks, which always outperform recurrent networks (for times less than the network size) in the scenario of gaussian signals and noise [6] are no longer optimal for sparse input statistics. Finally, we exploited powerful tools from statistical mechanics to analytically compute memory curves as a function of signal sparsity and network integration time. Our theoretically computed curves matched reasonably well simulations of orthogonal neural networks. To our knowledge, these results represent the first theoretical calculations of short-term memory curves for sparse signals in neuronal networks.

We emphasize that we are not suggesting that biological neural systems use $L_1$ minimization to reconstruct past inputs. Instead we use $L_1$ minimization in this work simply as a theoretical tool to probe the memory capabilities of neural networks. However, neural implementations of $L_1$ minimization exist [19, 20], so if stimulus reconstruction were the goal of a neural system, reconstruction performance similar to what is reported here could be obtained in a neurally plausible manner. Also, we found that orthogonal neural networks, because of their eigenvalue spectrum, display remarkable memory properties, similar to that of an annealed approximation. Such special connectivity is essential for memory performance, as random gaussian networks cannot have memory similar to the annealed approximation. Orthogonal connectivity could be implemented in a biologically plausible manner using antisymmetric networks with inhibition operating in continuous time. When exponentiated, such connectivities yield the orthogonal networks considered here in discrete time.

Our results are relevant not only to the field of short-term memory, but also to the field of compressed sensing (CS). We have introduced two new ensembles of random CS measurement matrices. The first of these, dynamical CS matrices, are the effective measurements a dynamical system makes on a continuous temporal stream of input. Dynamical CS matrices have three properties not considered in the existing CS literature: they are infinite in temporal extent, have columns that decay over time and exhibit correlations between columns. We also introduce annealed CS matrices, that are also infinite in extent and have decaying columns, but no correlations across columns. We show how to analytically calculate the time course of reconstruction error in the annealed ensemble and compare it to the dynamical ensemble for orthogonal dynamical systems. Our results show that orthogonal dynamical systems can perform CS even while operating with errors.

This work suggests several extensions. Given the importance of signal statistics in determining memory capacity, it would be interesting to study memory for sparse nonnegative signals. The inequality constraints on the space of allowed signals arising from nonnegativity can have important effects in CS; they shift the phase boundary between perfect and error-prone reconstruction [12, 13, 15], and they allow the existence of a new phase in which signal reconstruction is possible even without $L_1$ minimization [15]. We have found, through simulations, dramatic improvements in memory capacity in this case, and are extending the theory to explain these effects. Also, we have used a simple model for sparseness, in which a fraction of signal elements are nonzero. But our theory is general for any signal distribution, and could be used to analyze other models of sparsity, i.e. signals drawn from $L_p$ priors. Also, we have worked in the high SNR limit. However our theory can be extended to analyze memory in the presence of noise by working at finite $\lambda$. But most importantly, a deeper understanding of the relationship between dynamical CS matrices and their annealed counterparts would desirable. The effects of temporal correlations in the network activity patterns of orthogonal dynamical systems is central to this problem. For example, we have seen that these temporal correlations introduce strong correlations between the columns of the corresponding dynamical CS matrix (Fig. 2G), yet the memory properties of these matrices agree well with our annealed theory (Fig. 2E-F), which neglects these correlations. We leave this observation as an intriguing puzzle for the fields of short-term memory, dynamical systems, and compressed sensing.

**Acknowledgments**

S. G. and H. S. thank the Swartz Foundation, Burroughs Wellcome Fund, and the Israeli Science Foundation for support, and Daniel Lee for useful discussions.

# References

[1] J.J. Hopfield. Neural networks and physical systems with emergent collective computational abilities. *PNAS*, 79(8):2554, 1982.

[2] W. Maass, T. Natschlager, and H. Markram. Real-time computing without stable states: A new framework for neural computation based on perturbations. *Neural computation*, 14(11):2531–2560, 2002.

[3] H. Jaeger and H. Haas. Harnessing nonlinearity: Predicting chaotic systems and saving energy in wireless communication. *Science*, 304(5667):78, 2004.

[4] H. Jaeger. Short term memory in echo state networks. *GMD Report 152 German National Research Center for Information Technology*, 2001.

[5] O.L. White, D.D. Lee, and H. Sompolinsky. Short-term memory in orthogonal neural networks. *Phys. Rev. Lett.*, 92(14):148102, 2004.

[6] S. Ganguli, D. Huh, and H. Sompolinsky. Memory traces in dynamical systems. *Proc. Natl. Acad. Sci.*, 105(48):18970, 2008.

[7] A.M. Bruckstein, D.L. Donoho, and M. Elad. From sparse solutions of systems of equations to sparse modeling of signals and images. *Siam Review*, 51(1):34–81, 2009.

[8] E. Candes and M. Wakin. An introduction to compressive sampling. *IEEE Sig. Proc. Mag.*, 25(2):21–30, 2008.

[9] D.L. Donoho and M. Elad. Optimally sparse representation in general (non-orthogonal) dictionaries via l1 minimization. *PNAS*, 100:2197–2202, 2003.

[10] E. Candes, J. Romberg, and T. Tao. Robust uncertainty principles: Exact signal reconstruction from highly incomplete frequency information. *IEEE Trans. Inf. Theory*, 52(2):489–509, 2006.

[11] E. Candes and T. Tao. Decoding by linear programming. *IEEE Trans. Inf. Theory*, 51:4203–4215, 2005.

[12] D.L. Donoho and J. Tanner. Sparse nonnegative solution of underdetermined linear equations by linear programming. *PNAS*, 102:9446–51, 2005.

[13] D.L. Donoho and J. Tanner. Neighborliness of randomly projected simplices in high dimensions. *PNAS*, 102:9452–7, 2005.

[14] Y. Kabashima, T. Wadayama, and T. Tanaka. A typical reconstruction limit for compressed sensing based on l p-norm minimization. *J. Stat. Mech.*, page L09003, 2009.

[15] S. Ganguli and H. Sompolinsky. Statistical mechanics of compressed sensing. *Phys. Rev. Lett.*, 104(18):188701, 2010.

[16] M. Mezard, G. Parisi, and M.A. Virasoro. *Spin glass theory and beyond*. World scientific Singapore, 1987.

[17] S. Rangan, A.K. Fletcher, and Goyal V.K. Asymptotic analysis of map estimation via the replica method and applications to compressed sensing. *CoRR*, abs/0906.3234, 2009.

[18] D.L. Donoho, A. Maleki, and A. Montanari. Message-passing algorithms for compressed sensing. *Proc. Natl. Acad. Sci.*, 106(45):18914, 2009.

[19] Y. Xia and M.S. Kamel. A cooperative recurrent neural network for solving l 1 estimation problems with general linear constraints. *Neural computation*, 20(3):844–872, 2008.

[20] C.J. Rozell, D.H. Johnson, R.G. Baraniuk, and B.A. Olshausen. Sparse coding via thresholding and local competition in neural circuits. *Neural computation*, 20(10):2526–2563, 2008.

